# Efficient coding provides a direct link between prior and likelihood in perceptual Bayesian inference

**Xue-Xin Wei and Alan A. Stocker**\*
Departments of Psychology and
Electrical and Systems Engineering
University of Pennsylvania
Philadelphia, PA-19104, U.S.A.

## Abstract

A common challenge for Bayesian models of perception is the fact that the two fundamental Bayesian components, the prior distribution and the likelihood function, are formally unconstrained. Here we argue that a neural system that emulates Bayesian inference is naturally constrained by the way it represents sensory information in populations of neurons. More specifically, we show that an efficient coding principle creates a direct link between prior and likelihood based on the underlying stimulus distribution. The resulting Bayesian estimates can show biases *away* from the peaks of the prior distribution, a behavior seemingly at odds with the traditional view of Bayesian estimation, yet one that has been reported in human perception. We demonstrate that our framework correctly accounts for the repulsive biases previously reported for the perception of visual orientation, and show that the predicted tuning characteristics of the model neurons match the reported orientation tuning properties of neurons in primary visual cortex. Our results suggest that efficient coding is a promising hypothesis in constraining Bayesian models of perceptual inference.

## 1 Motivation

Human perception is not perfect. Biases have been observed in a large number of perceptual tasks and modalities, of which the most salient ones constitute many well-known perceptual illusions. It has been suggested, however, that these biases do not reflect a failure of perception but rather an observer's attempt to optimally combine the inherently noisy and ambiguous sensory information with appropriate prior knowledge about the world [13, 4, 14]. This hypothesis, which we will refer to as the *Bayesian hypothesis*, has indeed proven quite successful in providing a normative explanation of perception at a qualitative and, more recently, quantitative level (see *e.g.* [15]). A major challenge in forming models based on the Bayesian hypothesis is the correct selection of two main components: the prior distribution (belief) and the likelihood function. This has encouraged some to criticize the Bayesian hypothesis altogether, claiming that arbitrary choices for these components always allow for unjustified post-hoc explanations of the data [1].

We do not share this criticism, referring to a number of successful attempts to constrain prior beliefs and likelihood functions based on principled grounds. For example, prior beliefs have been defined as the relative distribution of the sensory variable in the environment in cases where these statistics are relatively easy to measure (*e.g.* local visual orientations [16]), or where it can be assumed that subjects have learned them over the course of the experiment (*e.g.* time perception [17]). Other studies have constrained the likelihood function according to known noise characteristics of neurons that are crucially involved in the specific perceptual process (*e.g* motion tuned neurons in visual cor-

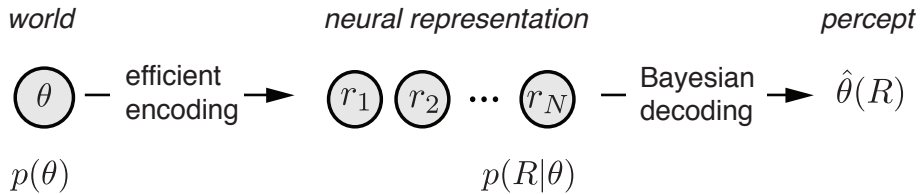

*world*        *neural representation*        *percept*

Figure 1: *Encoding-decoding framework.* A stimulus representing a sensory variable $\theta$ elicits a firing rate response $R = \{r_1, r_2, ..., r_N\}$ in a population of $N$ neurons. The perceptual task is to generate a good estimate $\hat{\theta}(R)$ of the presented value of the sensory variable based on this population response. Our framework assumes that encoding is efficient, and decoding is Bayesian based on the likelihood $p(R|\theta)$, the prior $p(\theta)$, and a squared-error loss function.

tex [18]). However, we agree that finding appropriate constraints is generally difficult and that prior beliefs and likelihood functions have been often selected on the basis of mathematical convenience.

Here, we propose that the *efficient coding hypothesis* [19] offers a joint constraint on the prior and likelihood function in neural implementations of Bayesian inference. Efficient coding provides a normative description of how neurons encode sensory information, and suggests a direct link between measured perceptual discriminability, neural tuning characteristics, and environmental statistics [11]. We show how this link can be extended to a full Bayesian account of perception that includes perceptual biases. We validate our model framework against behavioral as well as neural data characterizing the perception of visual orientation. We demonstrate that we can account not only for the reported perceptual biases *away* from the cardinal orientations, but also for the specific response characteristics of orientation-tuned neurons in primary visual cortex. Our work is a novel proposal of how two important normative hypotheses in perception science, namely efficient (en)coding and Bayesian decoding, might be linked.

## 2 Encoding-decoding framework

We consider perception as an inference process that takes place along the simplified neural encoding-decoding cascade illustrated in Fig. 1[1].

### 2.1 Efficient encoding

Efficient encoding proposes that the tuning characteristics of a neural population are adapted to the prior distribution $p(\theta)$ of the sensory variable such that the population optimally represents the sensory variable [19]. Different definitions of "optimally" are possible, and may lead to different results. Here, we assume an efficient representation that maximizes the mutual information between the sensory variable and the population response. With this definition and an upper limit on the total firing activity, the square-root of the Fisher Information must be proportional to the prior distribution [12, 21].

In order to constrain the tuning curves of individual neurons in the population we also impose a homogeneity constraint, requiring that there exists a one-to-one mapping $F(\theta)$ that transforms the physical space with units $\theta$ to a *homogeneous space* with units $\tilde{\theta} = F(\theta)$ in which the stimulus distribution becomes uniform. This defines the mapping as

$$F(\theta) = \int_{-\infty}^{\theta} p(\chi)d\chi \, , \tag{1}$$

which is the cumulative of the prior distribution $p(\theta)$. We then assume a neural population with identical tuning curves that evenly tiles the stimulus range in this homogeneous space. The population provides an efficient representation of the sensory variable $\theta$ according to the above constraints [11]. The tuning curves in the physical space are obtained by applying the inverse mapping $F^{-1}(\tilde{\theta})$. Fig. 2

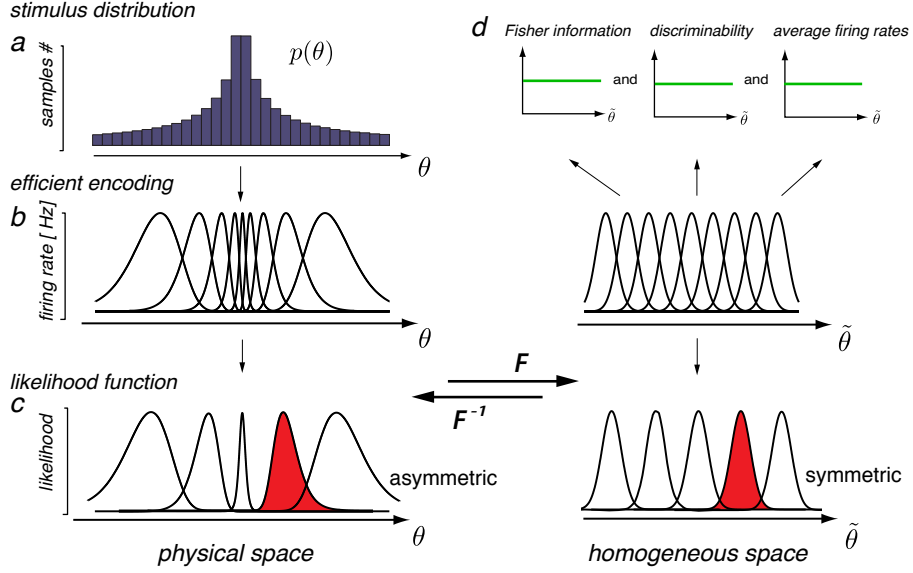

Figure 2: *Efficient encoding constrains the likelihood function.* a) Prior distribution $p(\theta)$ derived from stimulus statistics. b) Efficient coding defines the shape of the tuning curves in the physical space by transforming a set of homogeneous neurons using a mapping $F^{-1}$ that is the inverse of the cumulative of the prior $p(\theta)$ (see Eq. (1)). c) As a result, the likelihood shape is constrained by the prior distribution showing heavier tails on the side of lower prior density. d) Fisher information, discrimination threshold, and average firing rates are all uniform in the homogeneous space.

illustrates the applied efficient encoding scheme, the mapping, and the concept of the homogeneous space for the example of a symmetric, exponentially decaying prior distribution $p(\theta)$. The key idea here is that by assuming efficient encoding, the prior (*i.e.* the stimulus distribution in the world) directly constrains the likelihood function. In particular, the shape of the likelihood is determined by the cumulative distribution of the prior. As a result, the likelihood is generally asymmetric, as shown in Fig. 2, exhibiting heavier tails on the side of the prior with lower density.

## 2.2 Bayesian decoding

Let us consider a population of $N$ sensory neurons that efficiently represents a stimulus variable $\theta$ as described above. A stimulus $\theta_0$ elicits a specific population response that is characterized by the vector $R = [r_1, r_2, ..., r_N]$ where $r_i$ is the spike-count of the $i_{th}$ neuron over a given time-window $\tau$. Under the assumption that the variability in the individual firing rates is governed by a Poisson process, we can write the likelihood function over $\theta$ as

$$p(R|\theta) = \prod_{i=1}^{N} \frac{(\tau f_i(\theta))^{r_i}}{r_i!} e^{-\tau f_i(\theta)} , \tag{2}$$

with $f_i(\theta)$ describing the tuning curve of neuron $i$. We then define a Bayesian decoder $\hat{\theta}_{\text{LSE}}$ as the estimator that minimizes the expected squared-error between the estimate and the true stimulus value, thus

$$\hat{\theta}_{\text{LSE}}(R) = \frac{\int \theta p(R|\theta) p(\theta) d\theta}{\int p(R|\theta) p(\theta) d\theta} , \tag{3}$$

where we use Bayes' rule to appropriately combine the sensory evidence with the stimulus prior $p(\theta)$.

## 3 Bayesian estimates can be biased *away* from prior peaks

Bayesian models of perception typically predict perceptual biases *toward* the peaks of the prior density, a characteristic often considered a hallmark of Bayesian inference. This originates from the

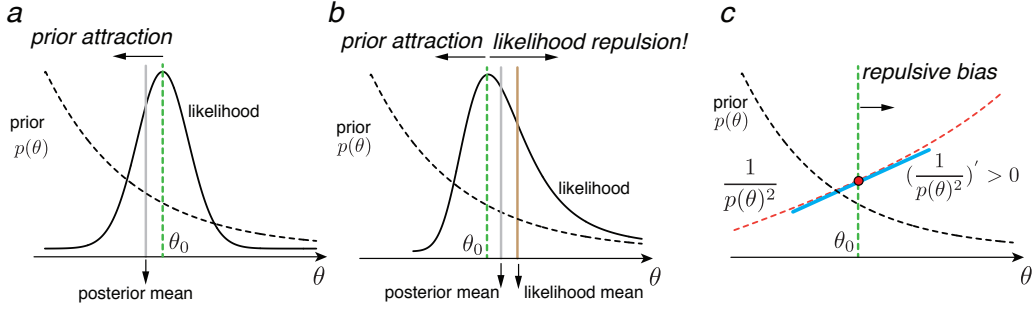

Figure 3: *Bayesian estimates biased **away** from the prior*. a) If the likelihood function is symmetric, then the estimate (posterior mean) is, on average, shifted away from the actual value of the sensory variable $\theta_0$ towards the prior peak. b) Efficient encoding typically leads to an asymmetric likelihood function whose normalized mean is away from the peak of the prior (relative to $\theta_0$). The estimate is determined by a combination of prior attraction and shifted likelihood mean, and can exhibit an overall repulsive bias. c) If $p(\theta_0)' < 0$ and the likelihood is relatively narrow, then $(1/p(\theta)^2)' > 0$ (blue line) and the estimate is biased away from the prior peak (see Eq. (6)).

common approach of choosing a parametric description of the likelihood function that is computationally convenient (*e.g.* Gaussian). As a consequence, likelihood functions are typically assumed to be symmetric (but see [23, 24]), leaving the bias of the Bayesian estimator to be mainly determined by the shape of the prior density, *i.e.* leading to biases toward the peak of the prior (Fig. 3a).

In our model framework, the shape of the likelihood function is constrained by the stimulus prior via efficient neural encoding, and is generally not symmetric for non-flat priors. It has a heavier tail on the side with lower prior density (Fig. 3b). The intuition is that due to the efficient allocation of neural resources, the side with smaller prior density will be encoded less accurately, leading to a broader likelihood function on that side. The likelihood asymmetry pulls the Bayes' least-squares estimate away from the peak of the prior while at the same time the prior pulls it toward its peak. Thus, the resulting estimation bias is the combination of these two counter-acting forces - and both are determined by the prior!

## 3.1   General derivation of the estimation bias

In the following, we will formally derive the mean estimation bias $b(\theta)$ of the proposed encoding-decoding framework. Specifically, we will study the conditions for which the bias is repulsive *i.e.* away from the peak of the prior density.

We first re-write the estimator $\hat{\theta}_{\text{LSE}}$ (3) by replacing $\theta$ with the inverse of its mapping to the homogeneous space, *i.e.*, $\theta = F^{-1}(\tilde{\theta})$. The motivation for this is that the likelihood in the homogeneous space is symmetric (Fig. 2). Given a value $\theta_0$ and the elicited population response $R$, we can write the estimator as

$$\hat{\theta}_{\text{LSE}}(R) = \frac{\int \theta p(R|\theta)p(\theta)d\theta}{\int p(R|\theta)p(\theta)d\theta} = \frac{\int F^{-1}(\tilde{\theta})p(R|F^{-1}(\tilde{\theta}))p(F^{-1}(\tilde{\theta}))dF^{-1}(\tilde{\theta})}{\int p(R|F^{-1}(\tilde{\theta}))p(F^{-1}(\tilde{\theta}))dF^{-1}(\tilde{\theta})}.$$

Calculating the derivative of the inverse function and noting that $F$ is the cumulative of the prior density, we get

$$dF^{-1}(\tilde{\theta}) = (F^{-1}(\tilde{\theta}))'d\tilde{\theta} = \frac{1}{F(\theta)'}d\tilde{\theta} = \frac{1}{p(\theta)}d\tilde{\theta} = \frac{1}{p(F^{-1}(\tilde{\theta}))}d\tilde{\theta}.$$

Hence, we can simplify $\hat{\theta}_{\text{LSE}}(R)$ as

$$\hat{\theta}_{\text{LSE}}(R) = \frac{\int F^{-1}(\tilde{\theta})p(R|F^{-1}(\tilde{\theta}))d\tilde{\theta}}{\int p(R|F^{-1}(\tilde{\theta}))d\tilde{\theta}}.$$

With

$$K(R, \tilde{\theta}) = \frac{p(R|F^{-1}(\tilde{\theta}))}{\int p(R|F^{-1}(\tilde{\theta}))d\tilde{\theta}}$$

we can further simplify the notation and get

$$\hat{\theta}_{\text{LSE}}(R) = \int F^{-1}(\tilde{\theta}) K(R, \tilde{\theta}) d\tilde{\theta} . \tag{4}$$

In order to get the expected value of the estimate, $\hat{\theta}_{\text{LSE}}(\tilde{\theta})$, we marginalize (4) over the population response space $S$,

$$\hat{\theta}_{\text{LSE}}(\tilde{\theta}) = \int_S \int p(R) F^{-1}(\tilde{\theta}) K(R, \tilde{\theta}) d\tilde{\theta} dR$$

$$= \int F^{-1}(\tilde{\theta}) (\int_S p(R) K(R, \tilde{\theta}) dR) d\tilde{\theta} = \int F^{-1}(\tilde{\theta}) L(\tilde{\theta}) d\tilde{\theta},$$

where we define

$$L(\tilde{\theta}) = \int_S p(R) K(R, \tilde{\theta}) dR.$$

It follows that $\int L(\tilde{\theta}) d\tilde{\theta} = 1$. Due to the symmetry in this space, it can be shown that $L(\tilde{\theta})$ is symmetric around the true stimulus value $\tilde{\theta}_0$. Intuitively, $L(\tilde{\theta})$ can be thought as the normalized *average likelihood* in the homogeneous space. We can then compute the expected bias at $\theta_0$ as

$$b(\theta_0) = \int F^{-1}(\tilde{\theta}) L(\tilde{\theta}) d\tilde{\theta} - F^{-1}(\tilde{\theta}_0) \tag{5}$$

This is expression is general where $F^{-1}(\tilde{\theta})$ is defined as the inverse of the cumulative of an arbitrary prior density $p(\theta)$ (see Eq. (1)) and the dispersion of $L(\tilde{\theta})$ is determined by the internal noise level.

Assuming the prior density to be smooth, we expand $F^{-1}$ in a neighborhood $(\tilde{\theta}_0 - h, \tilde{\theta}_0 + h)$ that is larger than the support of the likelihood function. Using Taylor's theorem with mean-value forms of the remainder, we get

$$F^{-1}(\tilde{\theta}) = F^{-1}(\tilde{\theta}_0) + F^{-1}(\tilde{\theta}_0)'(\tilde{\theta} - \tilde{\theta}_0) + \frac{1}{2} F^{-1}(\tilde{\theta}_x)''(\tilde{\theta} - \tilde{\theta}_0)^2 ,$$

with $\tilde{\theta}_x$ lying between $\tilde{\theta}_0$ and $\tilde{\theta}$. By applying this expression to (5), we find

$$b(\theta_0) = \int_{\tilde{\theta}_0 - h}^{\tilde{\theta}_0 + h} \frac{1}{2} F^{-1}(\tilde{\theta}_x)''(\tilde{\theta} - \tilde{\theta}_0)^2 L(\tilde{\theta}) d\tilde{\theta} = \frac{1}{2} \int_{\tilde{\theta}_0 - h}^{\tilde{\theta}_0 + h} (\frac{1}{p(F^{-1}(\tilde{\theta}_x))})'_{\tilde{\theta}} (\tilde{\theta} - \tilde{\theta}_0)^2 L(\tilde{\theta}) d\tilde{\theta}$$

$$= \frac{1}{2} \int_{\tilde{\theta}_0 - h}^{\tilde{\theta}_0 + h} -(\frac{p(\theta_x)'_\theta}{p(\theta_x)^3})(\tilde{\theta} - \tilde{\theta}_0)^2 L(\tilde{\theta}) d\tilde{\theta} = \frac{1}{4} \int_{\tilde{\theta}_0 - h}^{\tilde{\theta}_0 + h} (\frac{1}{p(\theta_x)^2})'_\theta (\tilde{\theta} - \tilde{\theta}_0)^2 L(\tilde{\theta}) d\tilde{\theta}.$$

In general, there is no simple rule to judge the sign of $b(\theta_0)$. However, if the prior is monotonic on the interval $F^{-1}((\tilde{\theta}_0 - h, \tilde{\theta}_0 + h))$, then the sign of $(\frac{1}{p(\theta_x)^2})'$ is always the same as the sign of $(\frac{1}{p(\theta_0)^2})'$. Also, if the likelihood is sufficiently narrow we can approximate $(\frac{1}{p(\theta_x)^2})'$ by $(\frac{1}{p(\theta_0)^2})'$, and therefore approximate the bias as

$$b(\theta_0) \approx C(\frac{1}{p(\theta_0)^2})' , \tag{6}$$

where C is a positive constant.

The result is quite surprising because it states that as long as the prior is monotonic over the support of the likelihood function, the expected estimation bias is *always away* from the peaks of the prior!

## 3.2   Internal (neural) versus external (stimulus) noise

The above derivation of estimation bias is based on the assumption that all uncertainty about the sensory variable is caused by neural response variability. This level of internal noise depends on the response magnitude, and thus can be modulated *e.g.* by changing stimulus contrast. This contrast-controlled noise modulation is commonly exploited in perceptual studies (*e.g.* [18]). Internal noise will always lead to repulsive biases in our framework if the prior is monotonic. If internal noise is low, the likelihood is narrow and thus the bias is small. Increasing internal noise leads to increasingly

larger biases up to the point where the likelihood becomes wide enough such that monotonicity of the prior over the support of the likelihood is potentially violated.

Stimulus noise is another way to modulate the noise level in perception (*e.g.* random-dot motion stimuli). Such external noise, however, has a different effect on the shape of the likelihood function as compared to internal noise. It modifies the likelihood function (2) by convolving it with the noise kernel. External noise is frequently chosen as additive and symmetric (*e.g.* zero-mean Gaussian). It is straightforward to prove that such symmetric external noise does not lead to a change in the mean of the likelihood, and thus does not alter the repulsive effect induced by its asymmetry. However, by increasing the overall width of the likelihood, the attractive influence of the prior increases, resulting in an estimate that is closer to the prior peak than without external noise[2].

# 4   Perception of visual orientation

We tested our framework by modelling the perception of visual orientation. Our choice was based on the fact that i) we have pretty good estimates of the prior distribution of local orientations in natural images, ii) tuning characteristics of orientation selective neurons in visual cortex are well-studied (monkey/cat), and iii) biases in perceived stimulus orientation have been well characterized. We start by creating an efficient neural population based on measured prior distributions of local visual orientation, and then compare the resulting tuning characteristics of the population and the predicted perceptual biases with reported data in the literature.

## 4.1   Efficient neural model population for visual orientation

Previous studies measured the statistics of the local orientation in large sets of natural images and consistently found that the orientation distribution is multimodal, peaking at the two cardinal orientations as shown in Fig. 4a [16, 20]. We assumed that the visual system's prior belief over orientation $p(\theta)$ follows this distribution and approximate it formally as

$$p(\theta) \propto 2 - |\sin(\theta)| \quad \text{(black line in Fig. 4b) .} \tag{7}$$

Based on this prior distribution we defined an efficient neural representation for orientation. We assumed a population of model neurons ($N = 30$) with tuning curves that follow a von-Mises distribution in the homogeneous space on top of a constant spontaneous firing rate (5 Hz). We then applied the inverse transformation $F^{-1}(\tilde{\theta})$ to all these tuning curves to get the corresponding tuning curves in the physical space (Fig. 4b - red curves), where $F(\theta)$ is the cumulative of the prior (7). The concentration parameter for the von-Mises tuning curves was set to $\kappa \approx 1.6$ in the homogeneous space in order to match the measured average tuning width ($\sim 32$ deg) of neurons in area V1 of the macaque [9].

## 4.2   Predicted tuning characteristics of neurons in primary visual cortex

The orientation tuning characteristics of our model population well match neurophysiological data of neurons in primary visual cortex (V1). Efficient encoding predicts that the distribution of neurons' preferred orientation follows the prior, with more neurons tuned to cardinal than oblique orientations by a factor of approximately 1.5. A similar ratio has been found for neurons in area V1 of monkey/cat [9, 10]. Also, the tuning widths of the model neurons vary between 25-42 deg depending on their preferred tuning (see Fig. 4c), matching the measured tuning width ratio of 0.6 between neurons tuned to the cardinal versus oblique orientations [9].

An important prediction of our model is that most of the tuning curves should be asymmetric. Such asymmetries have indeed been reported for the orientation tuning of neurons in area V1 [6, 7, 8]. We computed the *asymmetry index* for our model population as defined in previous studies [6, 7], and plotted it as a function of the preferred tuning of each neuron (Fig. 4d). The overall asymmetry index in our model population is $1.24 \pm 0.11$, which approximately matches the measured values for neurons in area V1 of the cat ($1.26 \pm 0.06$) [6]. It also predicts that neurons tuned to the cardinal and oblique orientations should show less symmetry than those tuned to orientations in between. Finally,

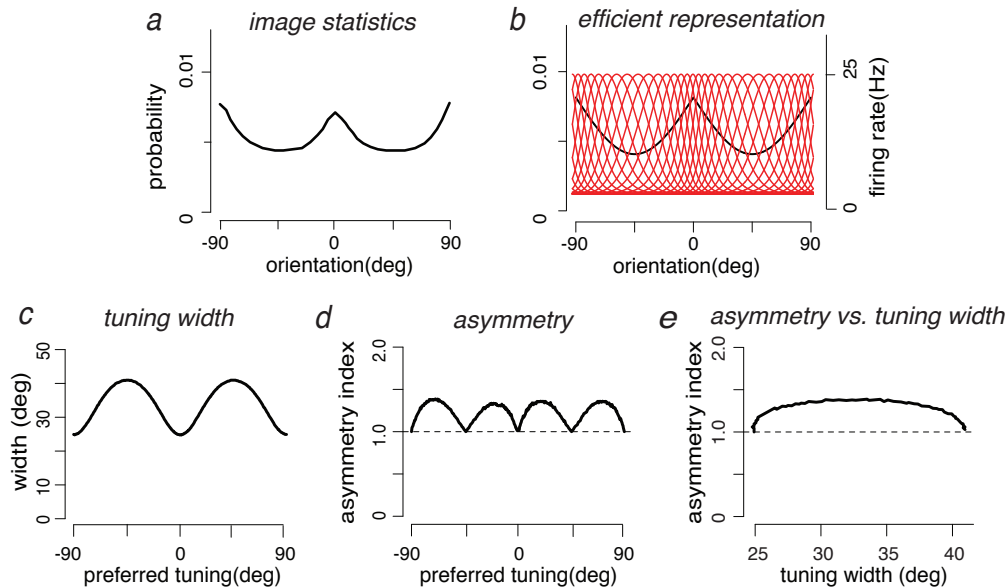

Figure 4: *Tuning characteristics of model neurons.* a) Distribution of local orientations in natural images, replotted from [16]. b) Prior used in the model (black) and predicted tuning curves according to efficient coding (red). c) Tuning width as a function of preferred orientation. d) Tuning curves of cardinal and oblique neurons are more symmetric than those tuned to orientations in between. e) Both narrowly and broadly tuned neurons neurons show less asymmetry than neurons with tuning widths in between.

neurons with tuning widths at the lower and upper end of the range are predicted to exhibit less asymmetry than those neurons whose widths lie in between these extremes (illustrated in Fig. 4e). These last two predictions have not been tested yet.

### 4.3  Predicted perceptual biases

Our model framework also provides specific predictions for the expected perceptual biases. Humans show systematic biases in perceived orientation of visual stimuli such as *e.g.* arrays of Gabor patches (Fig. 5a,d). Two types of biases can be distinguished: First, perceived orientations show an *absolute bias* away from the cardinal orientations, thus away from the peaks of the orientation prior [2, 3]. We refer to these biases as absolute because they are typically measured by adjusting a noise-free reference until it matched the orientation of the test stimulus. Interestingly, these repulsive absolute biases are the larger the smaller the external stimulus noise is (see Fig. 5b). Second, the *relative bias* between the perceived overall orientations of a high-noise and a low-noise stimulus is toward the cardinal orientations as shown in Fig. 5c, and thus toward the peak of the prior distribution [3, 16].

The predicted perceptual biases of our model are shown Fig. 5e,f. We computed the likelihood function according to (2) and used the prior in (7). External noise was modeled by convolving the stimulus likelihood function with a Gaussian (different widths for different noise levels). The predictions well match both, the reported absolute bias away as well as the relative biases toward the cardinal orientations. Note, that our model framework correctly accounts for the fact that less external noise leads to larger absolute biases (see also discussion in section 3.2).

## 5   Discussion

We have presented a modeling framework for perception that combines efficient (en)coding and Bayesian decoding. Efficient coding imposes constraints on the tuning characteristics of a population of neurons according to the stimulus distribution (prior). It thus establishes a direct link between prior and likelihood, and provides clear constraints on the latter for a Bayesian observer model of perception. We have shown that the resulting likelihoods are in general asymmetric, with

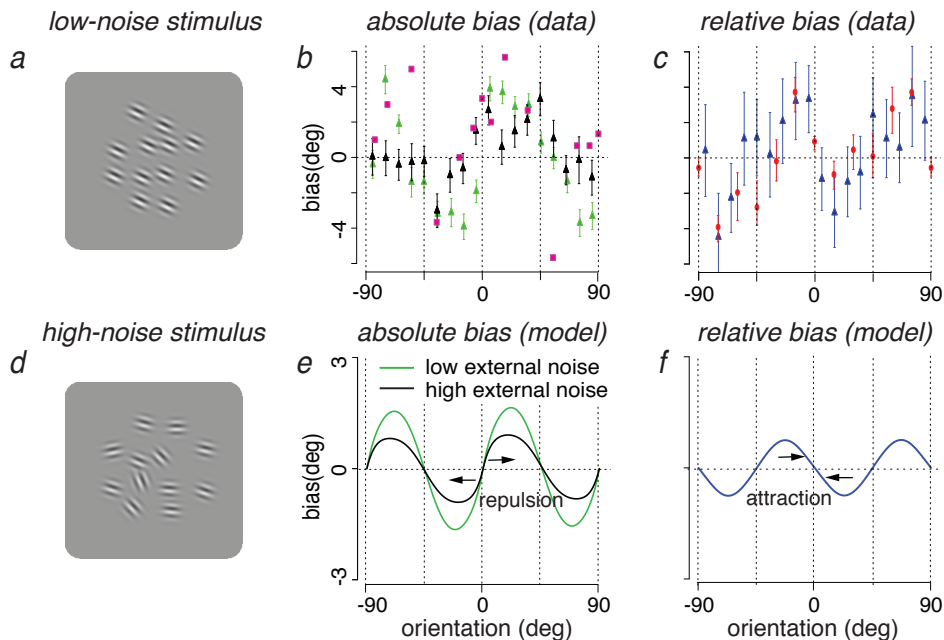

Figure 5: *Biases in perceived orientation: Human data vs. Model prediction.* a,d) Low- and high-noise orientation stimuli of the type used in [3, 16]. b) Humans show absolute biases in perceived orientation that are away from the cardinal orientations. Data replotted from [2] (pink squares) and [3] (green (black) triangles: bias for low (high) external noise). c) Relative bias between stimuli with different external noise level (high minus low). Data replotted from [3] (blue triangles) and [16] (red circles). e,f) Model predictions for absolute and relative bias.

heavier tails away from the prior peaks. We demonstrated that such asymmetric likelihoods can lead to the counter-intuitive prediction that a Bayesian estimator is biased *away* from the peaks of the prior distribution. Interestingly, such repulsive biases have been reported for human perception of visual orientation, yet a principled and consistent explanation of their existence has been missing so far. Here, we suggest that these counter-intuitive biases directly follow from the asymmetries in the likelihood function induced by efficient neural encoding of the stimulus. The good match between our model predictions and the measured perceptual biases and orientation tuning characteristics of neurons in primary visual cortex provides further support of our framework.

Previous work has suggested that there might be a link between stimulus statistics, neuronal tuning characteristics, and perceptual behavior based on efficient coding principles, yet none of these studies has recognized the importance of the resulting likelihood asymmetries [16, 11]. We have demonstrated here that such asymmetries can be crucial in explaining perceptual data, even though the resulting estimates appear "anti-Bayesian" at first sight (see also models of sensory adaptation [23]).

Note, that we do not provide a neural implementation of the Bayesian inference step. However, we and others have proposed various neural decoding schemes that can approximate Bayes' least-squares estimation using efficient coding [26, 25, 22]. It is also worth pointing out that our estimator is set to minimize total squared-error, and that other choices of the loss function (*e.g.* MAP estimator) could lead to different predictions. Our framework is general and should be directly applicable to other modalities. In particular, it might provide a new explanation for perceptual biases that are hard to reconcile with traditional Bayesian approaches [5].

## Acknowledgments

We thank M. Jogan and A. Tank for helpful comments on the manuscript. This work was partially supported by grant ONR N000141110744.

## Footnotes

\*http://www.sas.upenn.edu/ astocker/lab

[1]In the context of this paper, we consider 'inferring', 'decoding', and 'estimating' as synonymous.

[2]Note, that these predictions are likely to change if the external noise is not symmetric.

# References

[1] M. Jones, and B. C. Love. Bayesian fundamentalism or enlightenment? On the explanatory status and theoretical contributions of Bayesian models of cognition. *Behavioral and Brain Sciences*, 34, 169–231,2011.

[2] D. P. Andrews. Perception of contours in the central fovea. *Nature*, 205:1218- 1220, 1965.

[3] A. Tomassini, M. J.Morgam. and J. A. Solomon. Orientation uncertainty reduces perceived obliquity. *Vision Res*, 50, 541–547, 2010.

[4] W. S. Geisler, D. Kersten. Illusions, perception and Bayes. *Nature Neuroscience*, 5(6):508- 510, 2002.

[5] M. O. Ernst Perceptual learning: inverting the size-weight illusion. *Current Biology*, 19:R23- R25, 2009.

[6] G. H. Henry, B. Dreher, P. O. Bishop. Orientation specificity of cells in cat striate cortex. *J Neurophysiol*, 37(6):1394-409,1974.

[7] D. Rose, C. Blakemore An analysis of orientation selectivity in the cat's visual cortex. *Exp Brain Res.*, Apr 30;20(1):1-17, 1974.

[8] N. V. Swindale. Orientation tuning curves: empirical description and estimation of parameters. *Biol Cybern.*, 78(1):45-56, 1998.

[9] R. L. De Valois, E. W. Yund, N. Hepler. The orientation and direction selectivity of cells in macaque visual cortex. *Vision Res.*,22, 531544,1982.

[10] B. Li, M. R. Peterson, R. D. Freeman. The oblique effect: a neural basis in the visual cortex. *J. Neurophysiol.*, 90, 204217, 2003.

[11] D. Ganguli and E.P. Simoncelli. Implicit encoding of prior probabilities in optimal neural populations. In *Adv. Neural Information Processing Systems NIPS 23*, vol. 23:658–666, 2011.

[12] M. D. McDonnell, N. G. Stocks. Maximally Informative Stimuli and Tuning Curves for Sigmoidal Rate-Coding Neurons and Populations. *Phys Rev Lett.*, 101(5):058103, 2008.

[13] H Helmholtz. *Treatise on Physiological Optics (transl.)*. Thoemmes Press, Bristol, U.K., 2000. Original publication 1867.

[14] Y. Weiss, E. Simoncelli, and E. Adelson. Motion illusions as optimal percept. *Nature Neuroscience*, 5(6):598–604, June 2002.

[15] D.C. Knill and W. Richards, editors. *Perception as Bayesian Inference*. Cambridge University Press, 1996.

[16] A R Girshick, M S Landy, and E P Simoncelli. Cardinal rules: visual orientation perception reflects knowledge of environmental statistics. *Nat Neurosci*, 14(7):926–932, Jul 2011.

[17] M. Jazayeri and M.N. Shadlen. Temporal context calibrates interval timing. *Nature Neuroscience*, 13(8):914–916, 2010.

[18] A.A. Stocker and E.P. Simoncelli. Noise characteristics and prior expectations in human visual speed perception. *Nature Neuroscience*, pages 578–585, April 2006.

[19] H.B. Barlow. Possible principles underlying the transformation of sensory messages. In W.A. Rosenblith, editor, *Sensory Communication*, pages 217–234. MIT Press, Cambridge, MA, 1961.

[20] D.M. Coppola, H.R. Purves, A.N. McCoy, and D. Purves The distribution of oriented contours in the real world. *Proc Natl Acad Sci U S A.*, 95(7): 4002–4006, 1998.

[21] N. Brunel and J.-P. Nadal. Mutual information, Fisher information and population coding. *Neural Computation*, 10, 7, 1731–1757, 1998.

[22] X-X. Wei and A.A. Stocker. Bayesian inference with efficient neural population codes. In *Lecture Notes in Computer Science, Artificial Neural Networks and Machine Learning - ICANN 2012, Lausanne, Switzerland*, volume 7552, pages 523–530, 2012.

[23] A.A. Stocker and E.P. Simoncelli. Sensory adaptation within a Bayesian framework for perception. In Y. Weiss, B. Schölkopf, and J. Platt, editors, *Advances in Neural Information Processing Systems 18*, pages 1291–1298. MIT Press, Cambridge, MA, 2006. Oral presentation.

[24] D.C. Knill. Robust cue integration: A Bayesian model and evidence from cue-conflict studies with stereoscopic and figure cues to slant. *Journal of Vision*, 7(7):1–24, 2007.

[25] Deep Ganguli. Efficient coding and Bayesian inference with neural populations. *PhD thesis*, Center for Neural Science, New York University, New York, NY, September 2012.

[26] B. Fischer. Bayesian estimates from heterogeneous population codes. *In Proc. IEEE Intl. Joint Conf. on Neural Networks. IEEE*, 2010.

